# Envelope-based Planning in Relational MDPs

**Natalia H. Gardiol**
MIT AI Lab
Cambridge, MA 02139
nhg@ai.mit.edu

**Leslie Pack Kaelbling**
MIT AI Lab
Cambridge, MA 02139
lpk@ai.mit.edu

## Abstract

A mobile robot acting in the world is faced with a large amount of sensory data and uncertainty in its action outcomes. Indeed, almost all interesting sequential decision-making domains involve large state spaces and large, stochastic action sets. We investigate a way to act intelligently as quickly as possible in domains where finding a complete policy would take a hopelessly long time. This approach, Relational Envelope-based Planning (REBP) tackles large, noisy problems along two axes. First, describing a domain as a relational MDP (instead of as an atomic or propositionally-factored MDP) allows problem structure and dynamics to be captured compactly with a small set of probabilistic, relational rules. Second, an envelope-based approach to planning lets an agent begin acting quickly within a restricted part of the full state space and to judiciously expand its envelope as resources permit.

## 1 Introduction

Quickly generating generating usable plans when the world abounds with uncertainty is an important and difficult enterprise. Consider the classic blocks world domain: the number of ways to make a stack of a certain height grows exponentially with the number of blocks on the table; and if the outcomes of actions are uncertain, the task becomes even more daunting. We want planning techniques that can deal with large state spaces and large, stochastic action sets since most compelling, realistic domains have these characteristics. In this paper we propose a method for planning in very large domains by using expressive rules to restrict attention to high-utility subsets of the state space.

Much of the work in traditional planning techniques centers on propositional, deterministic domains. See Weld's survey [12] for an overview of the extensive work in this area. Efforts to extend classical planning approaches into stochastic domains include mainly techniques that work with fully-ground state spaces [13, 2]. Conversely, efforts to move beyond propositional STRIPS-based planning involve work in mainly deterministic domains [6, 10].

But the world is not deterministic: for an agent to act robustly, it must handle uncertain dynamics as well as large state and action spaces. Markov decision theory provides techniques for dealing with uncertain outcomes in atomic-state contexts, and much work has been done in leveraging structured representations to solve very large MDPs and some POMDPs [9, 3, 7]. While these techniques have moved MDP techniques from atomic-state representations to factored ones, they still operate in fully-ground state spaces.

In order to describe large stochastic domains compactly, we need relational structures that can represent uncertainty in the dynamics. Relational representations allow the structure of the domain to be expressed in terms of object *properties* rather than object identities and thus yield a much more compact representation of a domain than the equivalent propositional version can. Efficient solutions for probabilistic, first-order MDPs are difficult to come by, however. Boutilier *et al.*[3] find policies for first-order MDPs by solving for the value-function of a first-order domain: the approach manipulates logical expressions that stand for sets of underlying states, but keeping the value-function representation manageable requires complex theorem-proving. Other approaches in relational MDPs represent the value function as a decision-tree [5] or as a sum of local subfunctions [8]. Another recent body of work avoids learning the value function and learns policies directly from example policies [14]. These approaches all compute full policies over complete state and action spaces, however, and so are of a different spirit than the work presented here.

The underlying message is nevertheless clear: the more an agent can compute logically and the less it attends to particular domain objects, the more general its solutions will be. Since fully-ground representations grow too big to be useful and purely logical representations are as yet unwieldy, we propose a middle path: we agree to ground things out, but in a principled, restricted way. We represent world dynamics by a compact set of relational rules, and we extend the envelope method of Dean *et al.*[4] to use these structured dynamics. We quickly come up with an initial trajectory (an *envelope* of states) to the goal and then to refine the policy by gradually incorporating nearby states into the envelope. This approach avoids the wild growth of purely propositional techniques by restricting attention to a useful subset of states. Our approach strikes a balance along two axes: between fully ground and purely logical representations, and between straight-line plans and full MDP policies.

## 2   Planning with an Envelope in Relational Domains

The envelope method was initially designed for planning in atomic-state MDPs. Goals of achievement are encoded as reward functions, and planning now becomes finding a policy that maximizes a long-term measure of reward. Extending the approach to a relational setting lets us cast the problem of planning in stochastic, relational domains in terms of finding a policy for a restricted Markovian state space.

### 2.1   Encoding Markovian dynamics with rules

The first step to extending the envelope method to relational domains is to encode the world dynamics relationally. We use a compact set of rules, as in Figure 1. Each rule, or operator, is denoted by an action symbol and a parameterized argument list. Its behavior is defined by a precondition and a set of outcomes, together called the rule schema. Each precondition and outcome is a conjunction of domain predicates. A rule applies in a state if its precondition can be matched against some subset of the state ground predicates. Each outcome then describes the set of possible resulting ground states. Given this structured representation of action dynamics, we define a relational MDP as a tuple $\langle \mathcal{P}, \mathcal{Z}, \mathcal{O}, \mathcal{T}, R \rangle$:

*States:* The set of states is defined by a finite set $\mathcal{P}$ of relational predicates, representing the properties and relations that can hold among the finite set of domain objects, $\mathcal{O}$. Each RMDP state is a ground interpretation of the domain predicates over the domain objects.

*Actions:* The set of ground actions depends on the set of rules $\mathcal{Z}$ and the objects in the world. For example, move$(A, B)$ can be bound to the table arrangement in Figure 2(a) by binding $A$ to block 1 and $B$ to block 4 to yield the ground action move$(1, 4)$.

*Transition Dynamics:* For each action, the distribution over next states is given compactly by the distribution over outcomes encoded in the schema. For example, executing

move($A, B$)
*pre*: (clear($B$, t), hold(nil), height($B$,$H$), incr($H$,$H'$), clear($A$,t),on($A$,$C$),broke(f))
*eff*: $\begin{bmatrix} 0.70 \\ 0.30 \end{bmatrix}$ (on($A$, $B$), height($A$, $H$), clear($A$, t), clear($B$, f), hold(nil), clear($C$, t))
(on($A$, table), clear($A$, t), height($A$,$H$), hold(nil), clear($C$, t), broke(t))

fix()
*pre*: (broke(t))
*eff*: $\begin{bmatrix} 0.97 \\ 0.03 \end{bmatrix}$ (broke(f))
(broke(t))

stackon($B$)
*pre*: ( clear($B$, t), hold($A$), height($B$, $H$), incr($H$, $H'$), broke(f))
*eff*: $\begin{bmatrix} .97 \\ .03 \end{bmatrix}$ (on($A$, $B$), height($A$, $H$), clear($A$, t), clear(B, f),hold(nil))
(on($A$, table), clear($A$, t), height($A$,H'), hold(nil), broke(t))

stackon(table)
*pre*: (clear(table, t), hold($A$), broke(f))
*eff*: $[\, 1.00 \,]$ (on($A$, table), height($A$, 0), clear($A$, t), hold(nil))

pickup($A$)
*pre*: (clear($A$, t), hold(nil), on($A$, $B$),broke(f))
*eff*: $[\, 1.00 \,]$ (hold($A$), clear($A$, f), on($A$, nil), clear($B$, t), height($A$,-1))

Figure 1: The set of relational rules, $\mathcal{Z}$, for blocks-world dynamics.[2] Each rule schema contains the action name, precondition, and a set of effects.

move($1, 4$) yields a $0.3$ chance of landing in a state where block 1 falls on the table, and a $0.7$ chance of landing in a state where block 1 is correctly put on block 4. The rule outcomes themselves usually only specify a subset of the domain predicates, effectively describing a set of possible ground states. We assume a static frame: state predicates not directly changed by the rule are assumed to remain the same.

*Rewards:* A state is deterministically mapped to a scalar reward according to function $R(s)$.

## 2.2 Initial trajectory planning

The next step is finding an initial path. In a relational setting, when the underlying MDP space implied by the full instantiation of the representation is potentially huge, a good initial envelope is crucial. It determines the quality of the early envelope policies and sets the stage for more elaborate policies later on.

For planning in traditional STRIPS domains, the Graphplan algorithm is known to be effective [1]. Graphplan finds the shortest straight-line plan by iteratively growing a forward-chaining structure called a *plangraph* and testing for the presence of goal conditions at each step. Blum and Langford [2] describe a probabilistic extension called TGraphplan (TGP) that works by returning a plan's a probability of success rather than a just a boolean flag. TGP can find straight-line plans fairly quickly from start to goal that satisfy a minimum probability. Given TGP's success in probabilistic STRIPS domains, a straightforward idea is to use the trajectory found by TGP to populate our initial envelope.

Nevertheless, this should give us pause: we have just said that our relational MDP describes a large underlying MDP. TGP and other Graphplan descendants work by grounding out the rules and chaining them forward to construct the plangraph. Large numbers of actions cause severe problems for Graphplan-based planners [11] since the branching factor quickly chokes the forward-chaining plangraph construction. So how do we cope?

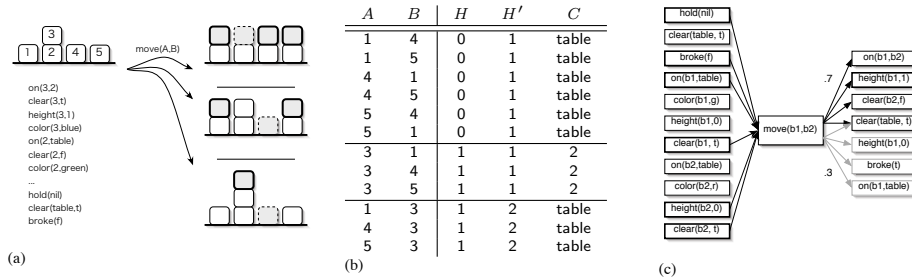

| A | B | H | H' | C |
|---|---|---|----|-------|
| 1 | 4 | 0 | 1 | table |
| 1 | 5 | 0 | 1 | table |
| 4 | 1 | 0 | 1 | table |
| 4 | 5 | 0 | 1 | table |
| 5 | 4 | 0 | 1 | table |
| 5 | 1 | 0 | 1 | table |
| 3 | 1 | 1 | 1 | 2 |
| 3 | 4 | 1 | 1 | 2 |
| 3 | 5 | 1 | 1 | 2 |
| 1 | 3 | 1 | 2 | table |
| 4 | 3 | 1 | 2 | table |
| 5 | 3 | 1 | 2 | table |

(a) (b) (c)

Figure 2: (a) Given this world configuration, the move action produces three types of effects. (b) 12 different groundings for the argument variables, but not all produce different groundings for the derived variables. (c) A plangraph fragment with a particular instance of move chained forward.

## 2.3 Equivalence-class sampling: reducing the planning action apace

STRIPS rules require every variable in the rule schema to appear in the argument list, so move$(A, B)$ becomes move$(A, B, H, H', C)$. The meaning of the operator shifts from "*move A onto B*" to "*move A at height H' onto B at height H from C*". Not only is this awkward, but specifying all the variables in the argument list yields an exponential number of ground actions as the number of domain objects grows. In contrast, the operators we defined above have argument lists containing only those variables that are *free* parameters. That is, when the operator move$(A, B)$ takes two arguments, $A$ and $B$, it means that the other variables (such as $C$, the block under $A$) are derivable from the relations in the rule schema. Guided by this observation, one can generalize among bindings that produce equivalent effects on the derivable properties.

Consider executing the move$(A, B)$ rule in the world configuration in Figure 2. This creates 12 fully-ground actions. However examining the bindings reveals only three types of action-effects. There is one group of actions that move a block from one block and onto another; a group that moves a block from the table and onto a block of height zero; and another group that moves a block off the table and onto a block of height one.

Except for the identities of the argument blocks $A$ and $B$, the actions in each class produce equivalent groundings for the properties of the related domain objects. Rather than using all the actions, then, the plangraph can be constructed chaining forward only a sampled action from each class. We call this *equivalence-class sampling*; the sampled action is *representative* of the effects of any action from that class. Sampling reduces the branching factor at each step in the plangraph, so significantly larger domains can be handled.

## 3 From a Planning Problem to a Policy

Now we describe the approach in detail. We define a planning problem as containing:

*Rules:* These are the relational operators that describe the action effects. In our system, they are designed by hand and the probabilities are specified by the programmer.

*Initial World State:* The set of ground predicates that describes the starting state. REBP does not make the closed world assumption, so all predicates and objects required in the planning task must appear in the initial state.

*Goal Condition:* A conjunction of relational predicates. The goal may contain variables — it does not need to be fully ground.

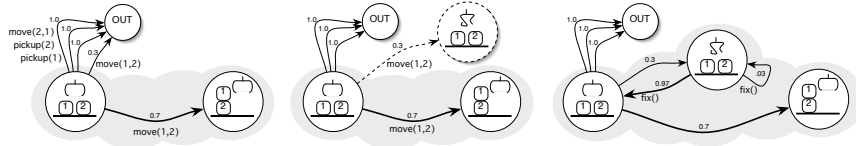

Figure 3: An initial envelope corresponding to the plangraph segment of Figure 2(c) followed fringe sampling and envelope expansion.

*Rewards:* A list of conjunctions mapping matching states to a scalar reward value. If a state in the current MDP does not match a reward condition, the default value is 0. Additionally, there must be a penalty associated with falling out of the envelope. This penalty is an estimate of the cost of having to recover from falling out (such as having to replan back to the envelope, for example).

Given a planning problem, there are now three main components to REBP: finding an initial plan, converting the plan into an MDP, and envelope manipulation. A running example to illustrate the approach will be the tiny task of making a two-block stack in a domain with two blocks. Figure 3 illustrates output produced by a run of the algorithm.

### 3.1 Finding an initial plan

The process for making the initial trajectory essentially follows the TGP algorithm described by Blum and Langford [2]. The TGP algorithm starts with the initial world state as the first layer in the graph, a minimum probability cutoff for the plan, and a maximum plan depth. We use the equivalence-class sampling technique discussed above to prune actions from the plangraph. Figure 2(c) shows one step of a plangraph construction.

### 3.2 Turning the initial plan into an MDP

The TGP algorithm produces a sequence of actions. The next step is to turn the sequence of action-effects into a well-defined envelope MDP; that is, we must compute the set of states and the transitions. Usually, the sequence of action-effects alone leaves many state predicates unspecified. Currently, we assume a static frame, which implies that the value of a predicate remains the same unless it is known to have explicitly changed.

The set of RMDP states are computed iteratively: first, the envelope is initialized with the initial world state; then, the next state in the envelope is found by applying the plan action to the previous state and "filling in" any missing predicates with their previous values; when the state containing the goal condition is reached, the set of states is complete. To compute the set of actions, REBP loops through the list of operators and accumulates all the ground actions whose preconditions bind to any state in the envelope. Transitions that initiate in an envelope state but do not land in an envelope state are redirected to OUT. The leftmost MDP in Figure 3 shows the initial envelope corresponding to the one-step plan of Figure 2(c).

### 3.3 Envelope Expansion

Envelope expansion, or *deliberation*, involves adding to the subset of world states under consideration. The decision of when and how long to deliberate must compaare the expected utility of further thinking against the cost of doing so. Dean *et al.* [4] discuss this complex issue in depth. As a first step, we considered the simple *precursor* deliberation model, in which deliberation occurs for some number $r$ times and is completed before execution takes place.

A round of deliberation involves sampling from the current policy to estimate which *fringe* states — states one step outside of the envelope — are likely. In each round, REBP draws $d \cdot M$ samples (drawing from an exploratory action with probability $\epsilon$) and keeps counts of which fringe states are reached. The $f \cdot M$ most likely fringes are added to the envelope, where $M$ is number of states in the current envelope and $d$ and $f$ are scalars. After expansion, we recompute the set of actions and compute a new policy.

Figure 3 shows a sequence of fringe sampling and envelope expansion. We see the incorporation of the fringe state in which the hand breaks as a result of move. With the new envelope, the policy is re-computed to include the fix action. This is a conditional plan that a straight-line planner could not find.

## 4 Experimental Domain

To illustrate the behavior of REBP, we show preliminary results in a stochastic blocks world. While simple, blocks world is a reasonably interesting first domain because, with enough blocks, it exposes the weaknesses of purely propositional approaches. Its regular dynamics, on other hand, lend themselves to relational descriptions. This domain demonstrates the type of scaling that can be achieved with the REBP approach.

The task at hand is to build a stack containing all the blocks on the table. In this domain, blocks are stacked *on* one another, with the top block in a stack being *clear*. Each block has a *color* and is at some *height* in the stack. There is a gripper that may or may not be *broken*. The pickup($A$) action is deterministic and puts a clear block into the empty hand; a block in the hand is no longer clear, and its height and and on-ness are no longer defined. The fix() action takes a broken hand and fixes it with some probability. The stackon() action comes in two flavors: first, stackon($B$), takes a block from the hand and puts it on block $B$, which may be dropped onto the table with a small probability; second, stackon(table), always puts the block from the hand onto the table. The move($A, B$) and stackon($B$) actions also have some chance of breaking the hand. If the hand is broken, it must be fixed before any further actions can apply. The domain is formalized as follows:[3]

$\mathcal{P}$ : on($Block, Block$), clear($Block, TorF$), color($Block, Color$),
height($Block, Num$), hold($Block$), clear(table, $TorF$), broke($TorF$).
$\mathcal{Z}, \mathcal{T}$ : The rules are shown in Figure 1.
$\mathcal{O}$ : A set of n differently colored (red, green, blue) blocks.
$R(s)$ : If $\exists A$ height($A, \mathsf{n} - 1$), then 1; if broke(t), then $-2$; if OUT, then $-1$.

## 5 Empirical Results

We compared the quality of the policies generated by the following algorithms: REBP; envelope expansion starting from empty initial plan (i.e., the initial envelope containing only the initial world state); and policy iteration on the fully ground MDP.[4]

In all cases, the policy was computed by simple policy iteration with a discount of $0.9$ and a stopping threshold of $0.1$. In the case of REBP, the number of deliberation rounds $r$ was 10, $d$ was 10, $f$ was 0.3, and $\epsilon$ was 0.2. In the case of the deliberation-only envelope, the $r$ was increased to 35. The runs were averaged over at least 7 trials in each case.

We show numerical results for domains with 5 and 6 blocks. The size of the full MDP in each case is, respectively, 768 and 5,228 states, with 351 and 733 ground actions. A

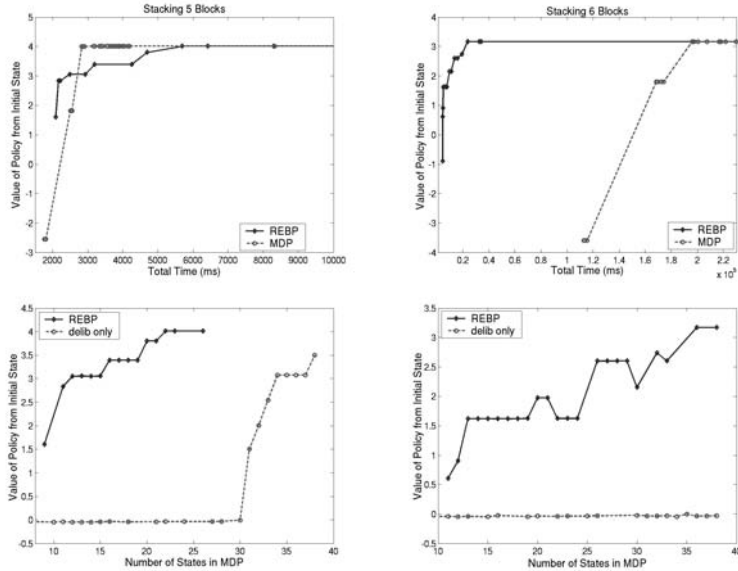

Figure 4: Results for the block-stacking tasks. The top plots show policy value against computation time for REBP and the full MDP. The bottom plots show policy value against number of states for REBP and deliberation only (empty initial plan).

domain of 7 blocks results in an MDP of over 37,000 states with 1,191 actions, a combined state and action space is too overwhelming for the full MDP solution. The REBP agent, on the other hand, is able to find plans for making stacks in domains of more than 12 blocks, which corresponds to an MDP of about 88,000 states and 3,000 ground actions.

The plots in Figure 4 show intuitive results. The top row shows the value of the policy against execution time (as measured by a monitoring package) showing that the REBP algorithm produces good quality plans quickly. For REBP, we start measuring the value of the policy at the point when initial trajectory finding ends and deliberation begins; for the full MDP solution, we measure the value of the policy at the end of each round of policy iteration. The full MDP takes a long time to find a policy, but eventually converges. Without the equivalence-class sampling, plangraph construction takes on the order of a couple of hours; with it, it takes a couple of minutes. The bottom row shows the value of the policy against the number of states in the envelope so far and shows that the a good initial envelope is key for behaving well with fewer states.

## 6 Discussion and Conclusions

Using the relational envelope method, we can take real advantage of relational generalization to produce good initial plans efficiently, and use envelope-growing techniques to improve the robustness of our plans incrementally as time permits. REBP is a planning system that tries to dynamically reformulate an apparently intractable problem into a small, easily handled problem at run time.

However, there is plenty remaining to be done. The first thing needed is a more rigorous analysis of the equivalence-class sampling. Currently, the action sampling is a purely local decision made at each step of the plangraph. This works in the current setup because object identities do not matter and properties not mentioned in the operator outcomes are never part of the goal condition. If, on the other hand, the goal was to make a stack of

height n − 1 with a green block on top, it could be problematic to construct the plangraph without considering block color in the sampled actions. We are currently investigating what conditions are necessary for making general guarantees about the sampling approach.

Furthermore, the current envelope-extension method is relatively undirected; it might be possible to diagnose more effectively which fringe states would be most profitable to add. In addition, techniques such as those used by Dean *et al.* [4] could be employed to decide when to stop envelope growth, and to manage the eventual interleaving of envelope-growth and execution. Currently the states in the envelope are essentially atomic; it ought to be possible to exploit the factored nature of relational representations to allow abstraction in the MDP model, with aggregate "states" in the MDP actually representing sets of states in the underlying world.

In summary, the REBP method provides a way to restrict attention to a small, useful subset of a large MDP space. It produces an initial plan quickly by taking advantage of generalization among action effects, and as a result behaves smarter in a large space much sooner than it could by waiting for a full solution.

## Acknowledgements

This work was supported by an NSF Graduate Research Fellowship, by the Office of Naval Research contract #N00014-00-1-0298, and by NASA award #NCC2-1237.

## Footnotes

[3]The predicates behave like functions in the sense that the $n^{th}$ argument represents the value of the relation for the first $n - 1$ arguments. Thus, we say clear(block5, f) instead of ¬clear(block5).

[4]Starting with the initial state, the set of states is generated by exhaustively applying our operators until no more new states are found; this yields the true set of reachable states.

## References

[1] Avrim L. Blum and Merrick L. Furst. Fast plannning through planning graph analysis. *Artificial Intelligence*, 90:281–300, 1997.

[2] Avrim L. Blum and John C. Langford. Probabilistic planning in the graphplan framework. In *5th European Conference on Planning*, 1999.

[3] Craig Boutilier, Raymond Reiter, and Bob Price. Symbolic dynamic programming for first-order MDPs. In *IJCAI*, 2001.

[4] Thomas Dean, Leslie Pack Kaelbling, Jak Kirman, and Ann Nicholson. Planning under time constraints in stochastic domains. *Artificial Intelligence*, 76, 1995.

[5] Kurt Driessens, Jan Ramon, and Hendrik Blockeel. Speeding up relational reinforcement learning through the use of an incremental first order decision tree learner. In *European Conference on Machine Learning*, 2001.

[6] B. Cenk Gazen and Craig A. Knoblock. Combining the expressivity of UCPOP with the efficiency of graphplan. In *Proc. European Conference on Planning (ECP-97)*, 1997.

[7] H. Geffner and B. Bonet. High-level planning and control with incomplete information using POMDPs. In *Fall AAAI Symposium on Cognitive Robotics*, 1998.

[8] C. Guestrin, D. Koller, C. Gearhart, and N. Kanodia. Generalizing plans to new environments in relational MDPs. In *International Joint Conference on Artificial Intelligence*, 2003.

[9] Jesse Hoey, Robert St-Aubin, Alan Hu, and Craig Boutilier. Spudd: Stochastic planning using decision diagrams. In *Fifteenth Conference on Uncertainty in Artificial Intelligence*, 1999.

[10] J. Koehler, B. Nebel, J. Hoffmann, and Y. Dimopoulos. Extending planning graphs to an ADL subset. In *Proc. European Conference on Planning (ECP-97)*, 1997.

[11] B. Nebel, J. Koehler, and Y. Dimopoulos. Ignoring irrelevant facts and operators in plan generation. In *Proc. European Conference on Planning (ECP-97)*, 1997.

[12] Daniel S. Weld. Recent advances in AI planning. *AI Magazine*, 20(2):93–123, 1999.

[13] Daniel S. Weld, Corin R. Anderson, and David E. Smith. Extending graphplan to handle uncertainty and sensing actions. In *Proceedings of AAAI '98*, 1998.

[14] SungWook Yoon, Alan Fern, and Robert Givan. Inductive policy selection for first-order MDPs. In *18th International Conference on Uncertainty in Artificial Intelligence*, 2002.
